# Generating more realistic images using gated MRF's

**Marc'Aurelio Ranzato**      **Volodymyr Mnih**      **Geoffrey E. Hinton**
Department of Computer Science
University of Toronto
{ranzato,vmnih,hinton}@cs.toronto.edu

## Abstract

Probabilistic models of natural images are usually evaluated by measuring performance on rather indirect tasks, such as denoising and inpainting. A more direct way to evaluate a generative model is to draw samples from it and to check whether statistical properties of the samples match the statistics of natural images. This method is seldom used with high-resolution images, because current models produce samples that are very different from natural images, as assessed by even simple visual inspection. We investigate the reasons for this failure and we show that by augmenting existing models so that there are two sets of latent variables, one set modelling pixel intensities and the other set modelling image-specific pixel covariances, we are able to generate high-resolution images that look much more realistic than before. The overall model can be interpreted as a gated MRF where both pair-wise dependencies and mean intensities of pixels are modulated by the states of latent variables. Finally, we confirm that if we disallow weight-sharing between receptive fields that overlap each other, the gated MRF learns more efficient internal representations, as demonstrated in several recognition tasks.

## 1   Introduction and Prior Work

The study of the statistical properties of natural images has a long history and has influenced many fields, from image processing to computational neuroscience [1]. In this work we focus on probabilistic models of natural images. These models are useful for extracting representations [2, 3, 4] that can be used for discriminative tasks and they can also provide adaptive priors [5, 6, 7] that can be used in applications like denoising and inpainting. Our main focus, however, will be on improving the quality of the generative model, rather than exploring its possible applications.

Markov Random Fields (MRF's) provide a very general framework for modelling natural images. In an MRF, an image is assigned a probability which is a normalized product of potential functions, with each function typically being defined over a subset of the observed variables. In this work we consider a very versatile class of MRF's in which potential functions are defined over both pixels and latent variables, thus allowing the states of the latent variables to modulate or *gate* the effective interactions between the pixels. This type of MRF, that we dub *gated MRF*, was proposed as an image model by Geman and Geman [8]. Welling et al. [9] showed how an MRF in this family[1] could be learned for small image patches and their work was extended to high-resolution images by Roth and Black [6] who also demonstrated its success in some practical applications [7].

Besides their practical use, these models were specifically designed to match the statistical properties of natural images, and therefore, it seems natural to evaluate them in those terms. Indeed, several authors [10, 7] have proposed that these models should be evaluated by generating images and

checking whether the samples match the statistical properties observed in natural images. It is, therefore, very troublesome that none of the existing models can generate good samples, especially for high-resolution images (see for instance fig. 2 in [7] which is one of the best models of high-resolution images reported in the literature so far). In fact, as our experiments demonstrate the generated samples from these models are more similar to random images than to natural images!

When MRF's with gated interactions are applied to small image patches, they actually seem to work moderately well, as demonstrated by several authors [11, 12, 13]. The generated patches have some coherent and elongated structure and, like natural image patches, they are predominantly very smooth with sudden outbreaks of strong structure. This is unsurprising because these models have a built-in assumption that images are very smooth with occasional strong violations of smoothness [8, 14, 15]. However, the extension of these patch-based models to high-resolution images by replicating filters across the image has proven to be difficult. The receptive fields that are learned no longer resemble Gabor wavelets but look random [6, 16] and the generated images lack any of the long range structure that is so typical of natural images [7]. The success of these methods in applications such as denoising is a poor measure of the quality of the generative model that has been learned: Setting the parameters to random values works almost as well for eliminating independent Gaussian noise [17], because this can be done quite well by just using a penalty for high-frequency variation.

In this work, we show that the generative quality of these models can be drastically improved by jointly modelling both pixel mean intensities and pixel covariances. This can be achieved by using two sets of latent variables, one that gates pair-wise interactions between pixels and another one that sets the mean intensities of pixels, as we already proposed in some earlier work [4]. Here, we show that this modelling choice is crucial to make the gated MRF work well on high-resolution images. Finally, we show that the most widely used method of sharing weights in MRF's for high-resolution images is overly constrained. Earlier work considered homogeneous MRF's in which each potential is replicated at all image locations. This has the subtle effect of making learning very difficult because of strong correlations at nearby sites. Following Gregor and LeCun [18] and also Tang and Eliasmith [19], we keep the number of parameters under control by using local potentials, but unlike Roth and Black [6] we only share weights between potentials that do not overlap.

## 2 Augmenting Gated MRF's with Mean Hidden Units

A Product of Student's t (PoT) model [15] is a gated MRF defined on small image patches that can be viewed as modelling image-specific, pair-wise relationships between pixel values by using the states of its latent variables. It is very good at representing the fact that two-pixel have very similar intensities and no good at all at modelling what these intensities are. Failure to model the mean also leads to impoverished modelling of the covariances when the input images have non-zero mean intensity. The covariance RBM (cRBM) [20] is another model that shares the same limitation since it only differs from PoT in the distribution of its latent variables: The posterior over the latent variables is a product of Bernoulli distributions instead of Gamma distributions as in PoT. We explain the fundamental limitation of these models by using a simple toy example: Modelling two-pixel images using a cRBM with only one binary hidden unit, see fig. 1.

This cRBM assumes that the conditional distribution over the input is a zero-mean Gaussian with a covariance that is determined by the state of the latent variable. Since the latent variable is binary, the cRBM can be viewed as a mixture of two zero-mean full covariance Gaussians. The latent variable uses the pairwise relationship between pixels to decide which of the two covariance matrices should be used to model each image. When the input data is pre-proessed by making each image have zero mean intensity (the empirical histogram is shown in the first row and first column), most images lie near the origin because most of the times nearby pixels are strongly correlated. Less frequently we encounter edge images that exhibit strong anti-correlation between the pixels, as shown by the long tails along the anti-diagonal line. A cRBM could model this data by using two Gaussians (first row and second column): one that is spherical and tight at the origin for smooth images and another one that has a covariance elongated along the anti-diagonal for structured images.

If, however, the whole set of images is normalized by subtracting from every pixel the mean value of all pixels over all images (second row and first column), the cRBM fails at modelling structured images (second row and second column). It can fit a Gaussian to the smooth images by discovering

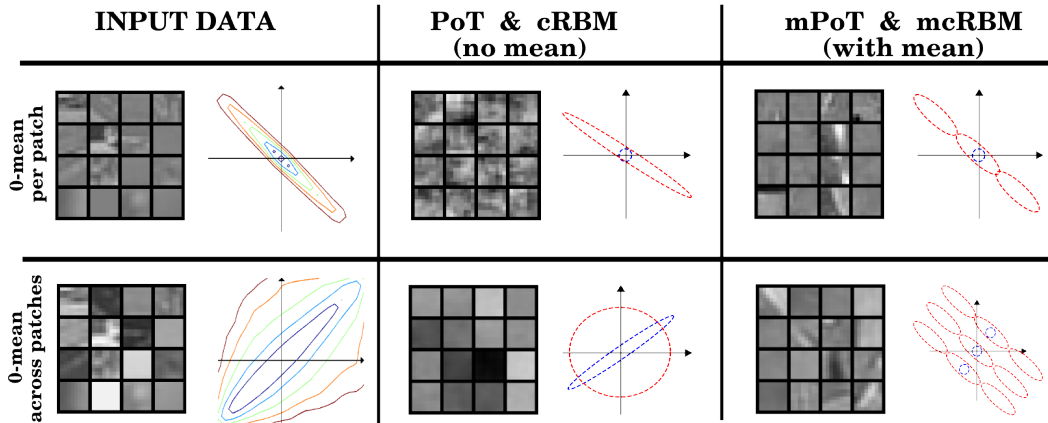

|  | INPUT DATA | PoT & cRBM (no mean) | mPoT & mcRBM (with mean) |

Figure 1: In the first row, each image is zero mean. In the second row, the whole set of data points is centered but each image can have non-zero mean. The first column shows 8x8 images picked at random from natural images. The images in the second column are generated by a model that does not account for mean intensity. The images in the third column are generated by a model that has both "mean" and "covariance" hidden units. The contours in the first column show the negative log of the empirical distribution of (tiny) natural two-pixel images (x-axis being the first pixel and the y-axis the second pixel). The plots in the other columns are toy examples showing how each model could represent the empirical distribution using a mixture of Gaussians with components that have one of two possible covariances (corresponding to the state of a binary "covariance" latent variable). Models that can change the means of the Gaussians (mPoT and mcRBM) can represent better structured images (edge images lie along the anti-diagonal and are fitted by the Gaussians shown in red) while the other models (PoT and cRBM) fail, overall when each image can have non-zero mean.

the direction of strong correlation along the main diagonal, but it is very likely to fail to discover the direction of anti-correlation, which is crucial to represent discontinuities, because structured images with different mean intensity appear to be evenly spread over the whole input space.

If the model has another set of latent variables that can change the means of the Gaussian distributions in the mixture (as explained more formally below and yielding the mPoT and mcRBM models), then the model can represent both changes of mean intensity and the correlational structure of pixels (see last column). The mean latent variables effectively subtract off the relevant mean from each data-point, letting the covariance latent variable capture the covariance structure of the data. As before, the covariance latent variable needs only to select between two covariance matrices.

In fact, experiments on real 8x8 image patches confirm these conjectures. Fig. 1 shows samples drawn from PoT and mPoT. mPoT (and similarly mcRBM [4]) is not only better at modelling zero mean images but it can also represent images that have non zero mean intensity well.

We now describe mPoT, referring the reader to [4] for a detailed description of mcRBM. In PoT [9] the energy function is:

$$E^{\mathbf{PoT}}(\mathbf{x}, \mathbf{h^c}) = \sum_i [h_i^c(1 + \frac{1}{2}(C_i^T \mathbf{x})^2) + (1 - \gamma) \log h_i^c] \qquad (1)$$

where $\mathbf{x}$ is a vectorized image patch, $\mathbf{h^c}$ is a vector of Gamma "covariance" latent variables, $C$ is a filter bank matrix and $\gamma$ is a scalar parameter. The joint probability over input pixels and latent variables is proportional to $\exp(-E^{\mathbf{PoT}}(\mathbf{x}, \mathbf{h^c}))$. Therefore, the conditional distribution over the input pixels is a *zero-mean* Gaussian with covariance equal to:

$$\Sigma^c = (C\text{diag}(\mathbf{h^c})C^T)^{-1}. \qquad (2)$$

In order to make the mean of the conditional distribution non-zero, we define mPoT as the normalized product of the above zero-mean Gaussian that models the covariance and a spherical covariance Gaussian that models the mean. The overall energy function becomes:

$$E^{\mathbf{mPoT}}(\mathbf{x}, \mathbf{h^c}, \mathbf{h^m}) = E^{\mathbf{PoT}}(\mathbf{x}, \mathbf{h^c}) + E^m(\mathbf{x}, \mathbf{h^m}) \qquad (3)$$

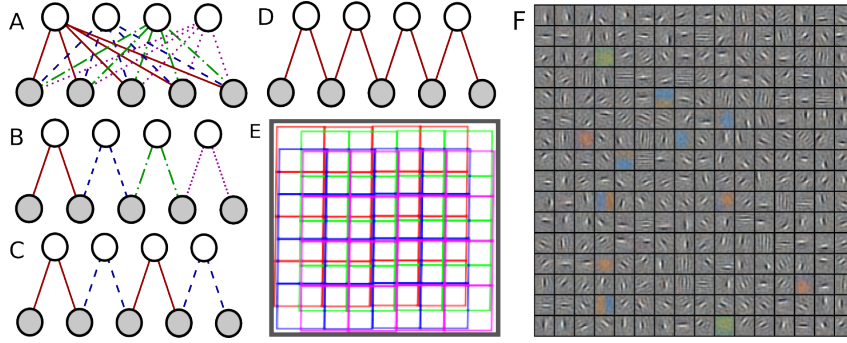

Figure 2: Illustration of different choices of weight-sharing scheme for a RBM. Links converging to one latent variable are filters. Filters with the same color share the same parameters. Kinds of weight-sharing scheme: **A)** *Global*, **B)** *Local*, **C)** *TConv* and **D)** *Conv*. **E)** *TConv* applied to an image. Cells correspond to neighborhoods to which filters are applied. Cells with the same color share the same parameters. **F)** 256 filters learned by a Gaussian RBM with *TConv* weight-sharing scheme on high-resolution natural images. Each filter has size 16x16 pixels and it is applied every 16 pixels in both the horizontal and vertical directions. Filters in position $(i, j)$ and $(1, 1)$ are applied to neighborhoods that are $(i, j)$ pixels away form each other. Best viewed in color.

where $\mathbf{h^m}$ is another set of latent variables that are assumed to be Bernoulli distributed (but other distributions could be used). The new energy term is:

$$E^m(\mathbf{x}, \mathbf{h^m}) = \frac{1}{2}\mathbf{x}^T\mathbf{x} - \sum_j h_j^m W_j^T \mathbf{x} \tag{4}$$

yielding the following conditional distribution over the input pixels:

$$p(\mathbf{x}|\mathbf{h^c}, \mathbf{h^m}) = N(\Sigma(W\mathbf{h^m}), \Sigma), \ \Sigma = (\Sigma^c + I)^{-1} \tag{5}$$

with $\Sigma^c$ defined in eq. 2. As desired, the conditional distribution has non-zero mean[2].

Patch-based models like PoT have been extended to high-resolution images by using spatially localized filters [6]. While we can subtract off the mean intensity from independent image patches to successfully train PoT, we cannot do that on a high-resolution image because overlapping patches might have different mean. Unfortunately, replicating potentials over the image ignoring variations of mean intensity has been the leading strategy to date [6][3]. This is the major reason why generation of high-resolution images is so poor. Sec. 4 shows that generation can be drastically improved by explicitly accounting for variations of mean intensity, as performed by mPoT and mcRBM.

## 3    Weight-Sharing Schemes

By integrating out the latent variables, we can write the density function of any gated MRF as a normalized product of potential functions (for mPoT refer to eq. 6). In this section we investigate different ways of constraining the parameters of the potentials of a generic MRF.
**Global:**   The obvious way to extend a patch-based model like PoT to high-resolution images is to define potentials over the whole image; we call this scheme *global*. This is not practical because 1) the number of parameters grows about quadratically with the size of the image making training too slow, 2) we do not need to model interactions between very distant pairs of pixels since their dependence is negligible, and 3) we would not be able to use the model on images of different size.
**Conv:**   The most popular way to handle big images is to define potentials on small subsets of variables (e.g., neighborhoods of size 5x5 pixels) and to replicate these potentials across space while

sharing their parameters at each image location [23, 24, 6]. This yields a *convolutional* weight-sharing scheme, also called homogeneous field in the statistics literature. This choice is justified by the stationarity of natural images. This weight-sharing scheme is extremely concise in terms of number of parameters, but also rather inefficient in terms of latent representation. First, if there are $N$ filters at each location and these filters are stepped by one pixel then the internal representation is about $N$ times overcomplete. The internal representation has not only high computational cost, but it is also highly redundant. Since the input is mostly smooth and the parameters are the same across space, the latent variables are strongly correlated as well. This inefficiency turns out to be particularly harmful for a model like PoT causing the learned filters to become "random" looking (see fig 3-iii). A simple intuition follows from the equivalence between PoT and square ICA [15]. If the filter matrix $C$ of eq. 1 is square and invertible, we can marginalize out the latent variables and write: $p(\mathbf{y}) = \prod_i S(y_i)$, where $y_i = C_i^T \mathbf{x}$ and $S$ is a Student's t distribution. In other words, there is an underlying assumption that filter outputs are independent. However, if the filters of matrix $C$ are shifted and overlapping versions of each other, this clearly cannot be true. Training PoT with the *Conv* weight-sharing scheme forces the model to find filters that make filter outputs as independent as possible, which explains the very high-frequency patterns that are usually discovered [6].

**Local:** The *Global* and *Conv* weight-sharing schemes are at the two extremes of a spectrum of possibilities. For instance, we can define potentials on a small subset of input variables but, unlike *Conv*, each potential can have its own set of parameters, as shown in fig. 2-B. This is called *local*, or inhomogeneous field. Compared to *Conv* the number of parameters increases only slightly but the number of latent variables required and their redundancy is greatly reduced. In fact, the model learns different receptive fields at different locations as a better strategy for representing the input, overall when the number of potentials is limited (see also fig. 2-F).

**TConv:** *Local* would not allow the model to be trained and tested on images of different resolution, and it might seem wasteful not to exploit the translation invariant property of images. We therefore advocate the use of a weight-sharing scheme that we call *tiled-convolutional* (*TConv*) shown in fig. 2-C and E [18]. Each filter tiles the image without overlaps with copies of itself (*i.e.* the stride equals the filter diameter). This reduces spatial redundancy of latent variables and allows the input images to have arbitrary size. At the same time, different filters do overlap with each other in order to avoid tiling artifacts. Fig. 2-F shows filters that were (jointly) learned by a Restricted Boltzmann Machine (RBM) [29] with Gaussian input variables using the *TConv* weight-sharing scheme.

## 4 Experiments

We train gated MRF's with and without mean hidden units using different weight-sharing schemes. The training procedure is very similar in all cases. We perform approximate maximum likelihood by using Fast Persistence Contrastive Divergence (FPCD) [25] and we draw samples by using Hybrid Monte Carlo (HMC) [26]. Since all latent variables can be exactly marginalized out we can use HMC on the free energy (negative logarithm of the marginal distribution over the input pixels). For mPoT this is:

$$F^{\text{mPoT}}(\mathbf{x}) = -\log(p(\mathbf{x})) + \text{const.} = \sum_{k,i} \gamma \log(1 + \frac{1}{2}(C_{ik}^T \mathbf{x}_k)^2) + \frac{1}{2}\mathbf{x}^T \mathbf{x} - \sum_{k,j} \log(1 + \exp(W_{jk}^T \mathbf{x}_k)) \quad (6)$$

where the index $k$ runs over spatial locations and $\mathbf{x}_k$ is the $k$-th image patch. FPCD keeps samples, called negative particles, that it uses to represent the model distribution. These particles are all updated after each weight update. For each mini-batch of data-points a) we compute the derivative of the free energy w.r.t. the training samples, b) we update the negative particles by running HMC for one HMC step consisting of 20 leapfrog steps. We start at the previous set of negative particles and use as parameters the sum of the regular parameters and a small perturbation vector, c) we compute the derivative of the free energy at the negative particles, and d) we update the regular parameters by using the difference of gradients between step a) and c) while the perturbation vector is updated using the gradient from c) only. The perturbation is also strongly decayed to zero and is subject to a larger learning rate. The aim is to encourage the negative particles to explore the space more quickly by slightly and temporarily raising the energy at their current position. Note that the use of FPCD as opposed to other estimation methods (like Persistent Contrastive Divergence [27]) turns out to be crucial to achieve good mixing of the sampler even after training. We train on mini-batches of 32 samples using gray-scale images of approximate size 160x160 pixels randomly cropped from the Berkeley segmentation dataset [28]. We perform 160,000 weight updates decreasing the learning by a factor of 4 by the end of training. The initial learning rate is set to 0.1 for the covariance

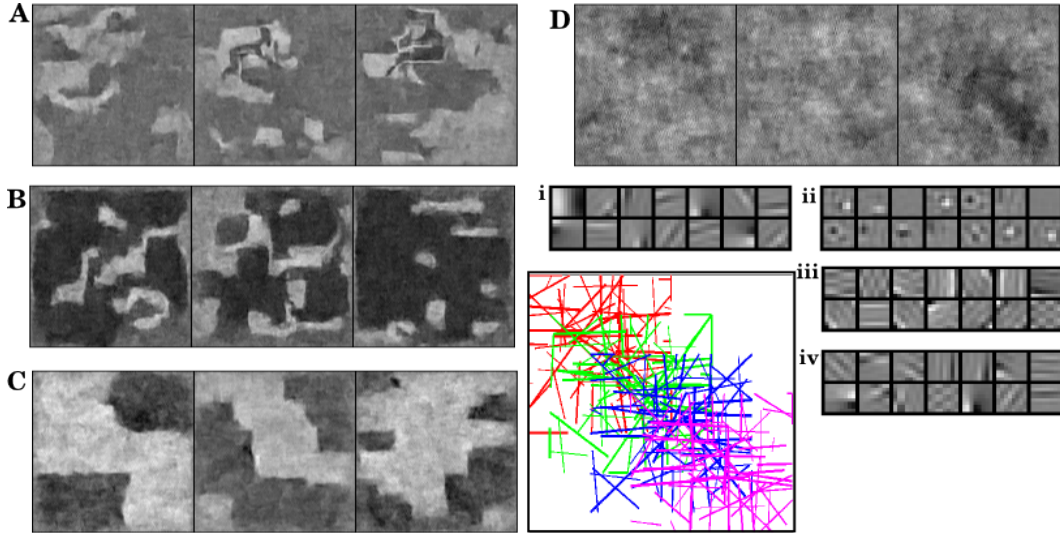

Figure 3: 160x160 samples drawn by A) mPoT-TConv, B) mHPoT-TConv, C) mcRBM-TConv and D) PoT-TConv. On the side also i) a subset of 8x8 "covariance" filters learned by mPoT-TConv (the plot below shows how the whole set of filters tile a small patch; each bar correspond to a Gabor fit of a filter and colors identify filters applied at the same 8x8 location, each group is shifted by 2 pixels down the diagonal and a high-resolution image is tiled by replicating this pattern every 8 pixels horizontally and vertically), ii) a subset of 8x8 "mean" filters learned by the same mPoT-TConv, iii) filters learned by PoT-Conv and iv) by PoT-TConv.

filters (matrix $C$ of eq. 1), 0.01 for the mean parameters (matrix $W$ of eq. 4), and 0.001 for the other parameters ($\gamma$ of eq. 1). During training we condition on the borders and initialize the negative particles at zero in order to avoid artifacts at the border of the image. We learn 8x8 filters and pre-multiply the covariance filters by a whitening transform retaining 99% of the variance; we also normalize the norm of the covariance filters to prevent some of them from decaying to zero during training[4].

Whenever we use the *TConv* weight-sharing scheme the model learns covariance filters that mostly resemble localized and oriented Gabor functions (see fig. 3-i and iv), while the *Conv* weight-sharing scheme learns structured but poorly localized high-frequency patterns (see fig. 3-iii) [6]. The *TConv* models re-use the same 8x8 filters every 8 pixels and apply a diagonal offset of 2 pixels between neighboring filters with different weights in order to reduce tiling artifacts. There are 4 sets of filters, each with 64 filters for a total of 256 covariance filters (see bottom plot of fig. 3). Similarly, we have 4 sets of mean filters, each with 32 filters. These filters have usually non-zero mean and exhibit on-center off-surround and off-center on-surround patterns, see fig. 3-ii.

In order to draw samples from the learned models, we run HMC for a long time (10,000 iterations, each composed of 20 leap-frog steps). Some samples of size 160x160 pixels are reported in fig. 3 A)-D). Without modelling the mean intensity, samples lack structure and do not seem much different from those that would be generated by a simple Gaussian model merely fitting the second order statistics (see fig. 3 in [1] and also fig. 2 in [7]). By contrast, structure, sharp boundaries and some simple texture emerge only from models that have mean latent variables, namely mcRBM, mPoT and mHPoT which differs from mPoT by having a second layer pooling matrix on the squared covariance filter outputs [11].

A more quantitative comparison is reported in table 1. We first compute marginal statistics of filter responses using the generated images, natural images from the test set, and random images. The statistics are the normalized histogram of individual filter responses to 24 Gabor filters (8 orientations and 3 scales). We then calculate the KL divergence between the histograms on random images and generated images and the KL divergence between the histograms on natural images and generated images. The table also reports the average difference of energies between random images and natural images. All results demonstrate that models that account for mean intensity generate images

| MODEL | $F(R) - F(T)\,(10^4)$ | $\mathrm{KL}(R \parallel G)$ | $\mathrm{KL}(T \parallel G)$ | $\mathrm{KL}(R \parallel G) - \mathrm{KL}(T \parallel G)$ |
|---|---|---|---|---|
| PoT - Conv | 2.9 | 0.3 | 0.6 | -0.3 |
| PoT - TConv | 2.8 | 0.4 | 1.0 | -0.6 |
| mPoT - TConv | **5.2** | 1.0 | 0.2 | 0.8 |
| mHPoT - TConv | 4.9 | 1.7 | 0.8 | **0.9** |
| mcRBM - TConv | 3.5 | 1.5 | 1.0 | 0.5 |

Table 1: Comparing MRF's by measuring: difference of energy (negative log ratio of probabilities) between random images (R) and test natural images (T), the KL divergence between statistics of random images (R) and generated images (G), KL divergence between statistics of test natural images (T) and generated images (G), and difference of these two KL divergences. Statistics are computed using 24 Gabor filters.

that are closer to natural images than to random images, whereas models that do not account for the mean (like the widely used PoT-Conv) produce samples that are actually closer to random images.

## 4.1 Discriminative Experiments on Weight-Sharing Schemes

In future work, we intend to use the features discovered by the generative model for recognition. To understand how the different weight sharing schemes affect recognition performance we have done preliminary tests using the discriminative performance of a simpler model on simpler data. We consider one of the simplest and most versatile models, namely the RBM [29]. Since we also aim to test the *Global* weight-sharing scheme we are constrained to using fairly low resolution datasets such as the MNIST dataset of handwritten digits [30] and the CIFAR 10 dataset of generic object categories [22]. The MNIST dataset has soft binary images of size 28x28 pixels, while the CIFAR 10 dataset has color images of size 32x32 pixels. CIFAR 10 has 10 classes, 5000 training samples per class and 1000 test samples per class. MNIST also has 10 classes with, on average, 6000 training samples per class and 1000 test samples per class.

The energy function of the RBM trained on the CIFAR 10 dataset, modelling input pixels with 3 (R,G,B) Gaussian variables [31], is exactly the one shown in eq. 4; while the RBM trained on MNIST uses logistic units for the pixels and the energy function is again the same as before but without any quadratic term. All models are trained in an unsupervised way to approximately maximize the likelihood in the training set using Contrastive Divergence [32]. They are then used to represent each input image with a feature vector (mean of the posterior over the latent variables) which is fed to a multinomial logistic classifier for discrimination. Models are compared in terms of: 1) recognition accuracy, 2) convergence time and 3) dimensionality of the representation. In general, assuming filters much smaller than the input image and assuming equal number of latent variables, *Conv*, *TConv* and *Local* models process each sample faster than *Global* by a factor approximately equal to the ratio between the area of the image and the area of the filters, which can be very large in practice.

In the first set of experiments reported on the left of fig. 4 we study the internal representation in terms of discrimination and dimensionality using the MNIST dataset. For each choice of dimensionality all models are trained using the same number of operations. This is set to the amount necessary to complete one epoch over the training set using the *Global* model. This experiment shows that: 1) *Local* outperforms all other weight-sharing schemes for a wide range of dimensionalities, 2) *TConv* does not perform as well as *Local* probably because the translation invariant assumption is clearly violated for these relatively small, centered, images, 3) *Conv* performs well only when the internal representation is very high dimensional (10 times overcomplete) otherwise it severely underfits, 4) *Global* performs well when the representation is compact but its performance degrades rapidly as this increases because it needs more than the allotted training time. The right hand side of fig. 4 shows how the recognition performance evolves as we increase the number of operations (or training time) using models that produce a twice overcomplete internal representation. With only very few filters *Conv* still underfits and it does not improve its performance by training for longer, but *Global* does improve and eventually it reaches the performance of *Local*. If we look at the crossing of the error rate at 2% we can see that *Local* is about 4 times faster than *Global*. To summarize, *Local* provides more compact representations than *Conv*, is much faster than *Global* while achieving

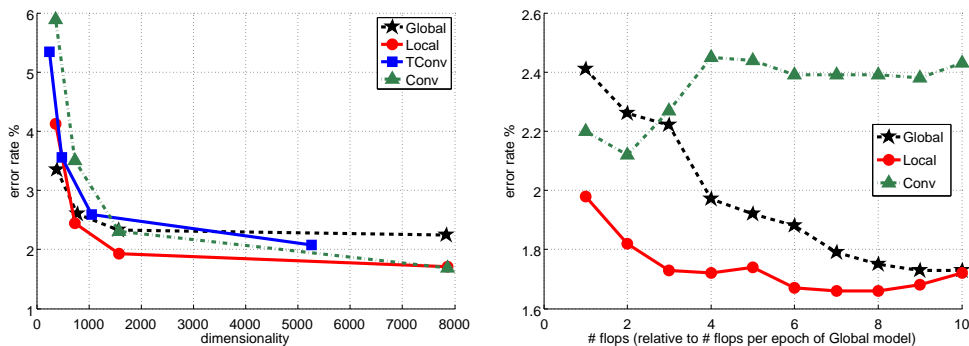

Figure 4: Experiments on MNIST using RBM's with different weight-sharing schemes. Left: Error rate as a function of the dimensionality of the latent representation. Right: Error rate as a function of the number of operations (normalized to those needed to perform one epoch in the *Global* model); all models have a twice overcomplete latent representation.

similar performance in discrimination. Also, *Local* can easily scale to larger images while *Global* cannot.

Similar experiments are performed using the CIFAR 10 dataset [22] of natural images. Using the same protocol introduced in earlier work by Krizhevsky [22], the RBM's are trained in an unsupervised way on a subset of the 80 million tiny images dataset [33] and then "fine-tuned" on the CIFAR 10 dataset by supervised back-propagation of the error through the linear classifier and feature extractor. All models produce an approximately 10,000 dimensional internal representation to make a fair comparison. Models using local filters learn 16x16 filters that are stepped every pixel. Again, we do not experiment with the *TConv* weight-sharing scheme because the image is not large enough to allow enough replicas.

Similarly to fig. 3-iii the *Conv* weight-sharing scheme was very difficult to train and did not produce Gabor-like features. Indeed, careful injection of sparsity and long training time seem necessary [31] for these RBM's. By contrast, both *Local* and *Global* produce Gabor-like filters similar to those shown in fig. 2 F). The model trained with *Conv* weight-sharing scheme yields an accuracy equal to 56.6%, while *Local* and *Global* yield much better performance, 63.6% and 64.8% [22], respectively. Although *Local* and *Global* have similar performance, training with the *Local* weight-sharing scheme took under an hour while using the *Global* weight-sharing scheme required more than a day.

## 5   Conclusions and Future Work

This work is motivated by the poor generative quality of currently popular MRF models of natural images. These models generate images that are actually more similar to white noise than to natural images. Our contribution is to recognize that current models can benefit from 1) the addition of a simple model of the mean intensities and from 2) the use of a less constrained weight-sharing scheme. By augmenting these models with an extra set of latent variables that model mean intensity we can generate samples that look much more realistic: they are characterized by smooth regions, sharp boundaries and some simple high frequency texture. We validate our approach by comparing the statistics of filter outputs on natural images and generated images.

In the future, we plan to integrate these MRF's into *deeper* hierarchical models and to use their internal representation to perform object recognition in high-resolution images. The hope is to further improve generation by capturing longer range dependencies and to exploit this to better cope with missing values and ambiguous sensory inputs.

## Footnotes

[1]Product of Student's t models (without pooling) may not appear to have latent variables but each potential can be viewed as an infinite mixture of zero-mean Gaussians where the inverse variance of the Gaussian is the latent variable.

[2]The need to model the means was clearly recognized in [21] but they used conjunctive latent features that simultaneously represented a contribution to the "precision matrix" in a specific direction and the mean along that same direction.

[3]The success of PoT-like models in Bayesian denoising is not surprising since the noisy image effectively replaces the reconstruction term from the mean hidden units (see eq. 5), providing a set of noisy mean intensities that are cleaned up by the patterns of correlation enforced by the covariance latent variables.

[4]The code used in the experiments can be found at the first author's web-page.

## References

[1] E.P. Simoncelli. Statistical modeling of photographic images. *Handbook of Image and Video Processing*, pages 431–441, 2005.

[2] A. Hyvarinen, J. Karhunen, and E. Oja. *Independent Component Analysis*. John Wiley & Sons, 2001.

[3] G.E. Hinton and R. R Salakhutdinov. Reducing the dimensionality of data with neural networks. *Science*, 313(5786):504–507, 2006.

[4] M. Ranzato and G.E. Hinton. Modeling pixel means and covariances using factorized third-order boltzmann machines. In *CVPR*, 2010.

[5] M.J. Wainwright and E.P. Simoncelli. Scale mixtures of gaussians and the statistics of natural images. In *NIPS*, 2000.

[6] S. Roth and M.J. Black. Fields of experts: A framework for learning image priors. In *CVPR*, 2005.

[7] U. Schmidt, Q. Gao, and S. Roth. A generative perspective on mrfs in low-level vision. In *CVPR*, 2010.

[8] S. Geman and D. Geman. Stochastic relaxation, gibbs distributions, and the bayesian restoration of images. *PAMI*, 6:721–741, 1984.

[9] M. Welling, G.E. Hinton, and S. Osindero. Learning sparse topographic representations with products of student-t distributions. In *NIPS*, 2003.

[10] S.C. Zhu and D. Mumford. Prior learning and gibbs reaction diffusion. *PAMI*, pages 1236–1250, 1997.

[11] S. Osindero, M. Welling, and G. E. Hinton. Topographic product models applied to natural scene statistics. *Neural Comp.*, 18:344–381, 2006.

[12] S. Osindero and G. E. Hinton. Modeling image patches with a directed hierarchy of markov random fields. In *NIPS*, 2008.

[13] Y. Karklin and M.S. Lewicki. Emergence of complex cell properties by learning to generalize in natural scenes. *Nature*, 457:83–86, 2009.

[14] B. A. Olshausen and D. J. Field. Sparse coding with an overcomplete basis set: a strategy employed by v1? *Vision Research*, 37:3311–3325, 1997.

[15] Y. W. Teh, M. Welling, S. Osindero, and G. E. Hinton. Energy-based models for sparse overcomplete representations. *JMLR*, 4:1235–1260, 2003.

[16] Y. Weiss and W.T. Freeman. What makes a good model of natural images? In *CVPR*, 2007.

[17] S. Roth and M. J. Black. Fields of experts. *Int. Journal of Computer Vision*, 82:205–229, 2009.

[18] K. Gregor and Y. LeCun. Emergence of complex-like cells in a temporal product network with local receptive fields. arXiv:1006.0448, 2010.

[19] C. Tang and C. Eliasmith. Deep networks for robust visual recognition. In *ICML*, 2010.

[20] M. Ranzato, A. Krizhevsky, and G.E. Hinton. Factored 3-way restricted boltzmann machines for modeling natural images. In *AISTATS*, 2010.

[21] N. Heess, C.K.I. Williams, and G.E. Hinton. Learning generative texture models with extended fields-of-experts. In *BMCV*, 2009.

[22] A. Krizhevsky. Learning multiple layers of features from tiny images, 2009. MSc Thesis, Dept. of Comp. Science, Univ. of Toronto.

[23] A. Waibel, T. Hanazawa, G. Hinton, K. Shikano, and K. Lang. Phoneme recognition using time-delay neural networks. *IEEE Acoustics Speech and Signal Proc.*, 37:328–339, 1989.

[24] Y. LeCun, L. Bottou, Y. Bengio, and P. Haffner. Gradient-based learning applied to document recognition. *Proceedings of the IEEE*, 86(11):2278–2324, 1998.

[25] T. Tieleman and G.E. Hinton. Using fast weights to improve persistent contrastive divergence. In *ICML*, 2009.

[26] R.M. Neal. *Bayesian learning for neural networks*. Springer-Verlag, 1996.

[27] T. Tieleman. Training restricted boltzmann machines using approximations to the likelihood gradient. In *ICML*, 2008.

[28] http://www.cs.berkeley.edu/projects/vision/grouping/segbench/.

[29] M. Welling, M. Rosen-Zvi, and G.E. Hinton. Exponential family harmoniums with an application to information retrieval. In *NIPS*, 2005.

[30] http://yann.lecun.com/exdb/mnist/.

[31] H. Lee, R. Grosse, R. Ranganath, and A. Y. Ng. Convolutional deep belief networks for scalable unsupervised learning of hierarchical representations. In *Proc. ICML*, 2009.

[32] G.E. Hinton. Training products of experts by minimizing contrastive divergence. *Neural Computation*, 14:1771–1800, 2002.

[33] A. Torralba, R. Fergus, and W.T. Freeman. 80 million tiny images: a large dataset for non-parametric object and scene recognition. *PAMI*, 30:1958–1970, 2008.

